# Deep Coding Network

**Yuanqing Lin**[†]    **Tong Zhang**[‡]    **Shenghuo Zhu**[†]    **Kai Yu**[†]
[†]NEC Laboratories America, Cupertino, CA 95129
[‡]Rutgers University, Piscataway, NJ 08854

## Abstract

This paper proposes a principled extension of the traditional single-layer flat sparse coding scheme, where a two-layer coding scheme is derived based on theoretical analysis of nonlinear functional approximation that extends recent results for local coordinate coding. The two-layer approach can be easily generalized to deeper structures in a hierarchical multiple-layer manner. Empirically, it is shown that the deep coding approach yields improved performance in benchmark datasets.

## 1   Introduction

Sparse coding has attracted significant attention in recent years because it has been shown to be effective for some classification problems [12, 10, 9, 13, 11, 14, 2, 5]. In particular, it has been empirically observed that *high-dimensional* sparse coding plus linear classifier is successful for image classification tasks such as PASCAL 2009 [7, 15].

The empirical success of sparse coding can be justified by theoretical analysis [17], which showed that a modification of sparse coding with added locality constraint, called local coordinate coding (LCC), represents a new class of effective high dimensional non-linear function approximation methods with sound theoretical guarantees. Specifically, LCC learns a nonlinear function in high dimension by forming an adaptive set of basis functions on the data manifold, and it has nonlinear approximation power. A recent extension of LCC with added local tangent directions [16] demonstrated the possibility to achieve locally quadratic approximation power when the underlying data manifold is relatively flat. This also indicates that the nonlinear function approximation view of sparse coding not only yields deeper theoretical understanding of its success, but also leads to improved algorithms based on refined analysis. This paper follows the same idea, where we propose a principled extension of single-layer sparse coding based on theoretical analysis of a two level coding scheme.

The algorithm derived from this approach has some advantages over the single-layer approach, and can also be extended into multi-layer hierarchical systems. Such extension draws connection to deep belief networks (DBN) [8], and hence we call this approach deep coding network. Hierarchical sparse coding has two main advantages over its single-layer counter-part. First, at the intuitive level, the first layer (traditional single-layer basis) yields a crude description of the data at each basis function, and multi-layer basis functions provide a natural way to zoom into each single basis for finer local details — this intuition can be reflected more rigorously in our nonlinear function approximation result. Due to the more localized zoom-in effect, it also alleviates the problem of overfitting when many basis functions are needed. Second, it is computationally more efficient than flat coding because we only need to look at locations in the second (or higher) layer corresponding to basis functions with nonzero coefficients in the first (or previous) layer. Since sparse coding produces many zero coefficients, the hierarchical structure significantly eliminates many of the coding computation. Moreover, instead of fitting a single model with many variables as in a flat single layer approach, our proposal of multi-layer coding requires fitting many small models separately, each

with a small number of parameters. In particular, fitting the small models can be done in parallel, e.g. using Hadoop, so that learning a fairly big number of codebooks can still be fast.

## 2 Sparse Coding and Nonlinear Function Approximation

This section reviews the nonlinear function approximation results of single-layer coding scheme in [17], and then presents our multi-layer extension. Since the result of [17] requires a modification of the traditional sparse coding scheme called local coordinate coding (LCC), our analysis will rely on a similar modification.

Consider the problem of learning a nonlinear function $f(x)$ in high dimension: $x \in \mathbb{R}^d$ with large $d$. While there are many algorithms in traditional statistics that can learn such a function in low dimension, when the dimensionality $d$ is large compared to $n$, the traditional statistical methods will suffer the so called "curse of dimensionality". The recently popularized coding approach addresses this issue. Specifically, it was theoretically shown in [17] that a specific coding scheme called Local Coordinate Coding can take advantage of the underlying data manifold geometric structure in order to learn a nonlinear function in high dimension and alleviate the curse of dimensionality problem.

The main idea of LCC, described in [17], is to locally embed points on the underlying data manifold into a lower dimensional space, expressed as coordinates with respect to a set of anchor points. The main theoretical observation was relatively simple: it was shown in [17] that on the data manifold, a nonlinear function can be effectively approximated by a globally linear function with respect to the local coordinate coding. Therefore the LCC approach turns a very difficult high dimensional nonlinear learning problem into a much simpler linear learning problem, which can be effectively solved using standard machine learning techniques such as regularized linear classifiers. This linearization is effective because the method naturally takes advantage of the geometric information.

In order to describe the results more formally, we introduce a number of notations. First we denote by $\|\cdot\|$ the Euclidean norm (2-norm) on $\mathbb{R}^d$:

$$\|x\| = \|x\|_2 = \sqrt{x_1^2 + \cdots + x_d^2}.$$

**Definition 2.1 (Smoothness Conditions)** *A function $f(x)$ on $\mathbb{R}^d$ is $(\alpha, \beta, \nu)$ Lipschitz smooth with respect to a norm $\|\cdot\|$ if*

$$\|\nabla f(x)\| \leq \alpha,$$

*and*

$$\left| f(x') - f(x) - \nabla f(x)^\top (x' - x) \right| \leq \beta \|x' - x\|^2,$$

*and*

$$\left| f(x') - f(x) - 0.5(\nabla f(x') + \nabla f(x))^\top (x' - x) \right|$$
$$\leq \nu \|x - x'\|^3,$$

*where we assume $\alpha, \beta, \nu \geq 0$.*

These conditions have been used in [16], and they characterize the smoothness of $f$ under zero-th, first, and second order approximations. The parameter $\alpha$ is the Lipschitz constant of $f(x)$, which is finite if $f(x)$ is Lipschitz; in particular, if $f(x)$ is constant, then $\alpha = 0$. The parameter $\beta$ is the Lipschitz derivative constant of $f(x)$, which is finite if the derivative $\nabla f(x)$ is Lipschitz; in particular, if $\nabla f(x)$ is constant (that is, $f(x)$ is a linear function of $x$), then $\beta = 0$. The parameter $\nu$ is the Lipschitz Hessian constant of $f(x)$, which is finite if the Hessian of $f(x)$ is Lipschitz; in particular, if the Hessian $\nabla^2 f(x)$ is constant (that is, $f(x)$ is a quadratic function of $x$), then $\nu = 0$. In other words, these parameters measure different levels of smoothness of $f(x)$: locally when $\|x - x'\|$ is small, $\alpha$ measures how well $f(x)$ can be approximated by a constant function, $\beta$ measures how well $f(x)$ can be approximated by a linear function in $x$, and $\nu$ measures how well $f(x)$ can be approximated by a quadratic function in $x$. For local constant approximation, the error term $\alpha \|x - x'\|$ is the first order in $\|x - x'\|$; for local linear approximation, the error term $\beta \|x - x'\|^2$ is the second order in $\|x - x'\|$; for local quadratic approximation, the error term $\nu \|x - x'\|^3$ is the third order in $\|x - x'\|$. That is, if $f(x)$ is smooth with relatively small $\alpha$, $\beta$, $\nu$, the error term becomes smaller (locally when $\|x - x'\|$ is small) if we use a higher order approximation.

Similar to the single-layer coordinate coding in [17], here we define a two-layer coordinate coding as the following.

**Definition 2.2 (Coordinate Coding)** *A single-layer coordinate coding is a pair $(\gamma^1, C^1)$, where $C^1 \subset \mathbb{R}^d$ is a set of anchor points (aka basis functions), and $\gamma$ is a map of $x \in \mathbb{R}^d$ to $[\gamma_v^1(x)]_{v \in C^1} \in \mathbb{R}^{|C^1|}$ such that $\sum_{v \in C^1} \gamma_v^1(x) = 1$. It induces the following physical approximation of $x$ in $\mathbb{R}^d$:*

$$h_{\gamma^1, C^1}(x) = \sum_{v \in C^1} \gamma_v^1(x) v.$$

*A two-layer coordinate coding $(\gamma, C)$ consists of coordinate coding systems $\{(\gamma^1, C^1)\} \cup \{(\gamma^{2,v}, C^{2,v}) : v \in C^1\}$. The pair $(\gamma^1, C^1)$ is the first layer coordinate coding, $(\gamma^{2,v}, C^{2,v})$ are second layer coordinate-coding pairs that refine the first layer coding for every first-layer anchor point $v \in C^1$.*

The performance of LCC is characterized in [17] using the following nonlinear function approximation result.

**Lemma 2.1 (Single-layer LCC Nonlinear Function Approximation)** *Let $(\gamma^1, C^1)$ be an arbitrary single-layer coordinate coding scheme on $\mathbb{R}^d$. Let $f$ be an $(\alpha, \beta, \nu)$-Lipschitz smooth function. We have for all $x \in \mathbb{R}^d$:*

$$\left| f(x) - \sum_{v \in C^1} w_v \gamma_v^1(x) \right| \le \alpha \left\| x - h_{\gamma^1, C^1}(x) \right\| + \beta \sum_{v \in C^1} |\gamma_v^1(x)| \| v - x \|^2, \tag{1}$$

*where $w_v = f(v)$ for $v \in C^1$.*

This result shows that a high dimensional nonlinear function can be globally approximated by a linear function with respect to the single-layer coding $[\gamma_v^1(x)]$, with unknown linear coefficients $[w_v]_{v \in C^1} = [f(v)]_{v \in C^1}$, where the approximation on the right hand size is second order. This bounds directly suggests the following learning method: for each $x$, we use its coding $[\gamma_v^1(x)] \in \mathbb{R}^{|C^1|}$ as features. We then learn a linear function of the form $\sum_v w_v \gamma_v^1(x)$ using a standard linear learning method such as SVM, where $[w_v]$ is the unknown coefficient vector to be learned. The optimal coding can be learned using unlabeled data by optimizing the right hand side of (1) over unlabeled data.

In the same spirit, we can extend the above result on LCC by including additional layers. This leads to the following bound.

**Lemma 2.2 (Two-layer LCC Nonlinear Function Approximation)** *Let $(\gamma, C) = \{(\gamma^1, C^1)\} \cup \{(\gamma^{2,v}, C^{2,v}) : v \in C^1\}$ be an arbitrary two-layer coordinate coding on $\mathbb{R}^d$. Let $f$ be an $(\alpha, \beta, \nu)$-Lipschitz smooth function. We have for all $x \in \mathbb{R}^d$:*

$$\| f(x) - \sum_{v \in C^1} w_v \gamma_v^1(x) - \sum_{v \in C^1} \gamma_v^1(x) \sum_{u \in C^{2,v}} w_{v,u} \gamma_u^{2,v}(x) \|$$

$$\le 0.5\alpha \| x - h_{\gamma^1, C^1}(x) \| + 0.5\alpha \sum_{v \in C^1} |\gamma_v^1(x)| \| x - h_{\gamma^{2,v}, C^{2,v}}(x) \| + \nu \sum_{v \in C^1} |\gamma_v^1(x)| \| x - v \|^3, \tag{2}$$

*where $w_v = f(v)$ for $v \in C^1$ and $w_{v,u} = 0.5 \nabla f(v)^\top (u - v)$ for $u \in C^{2,v}$, and*

$$\| f(x) - \sum_{v \in C^1} \gamma_v^1(x) \sum_{u \in C^{2,v}} w_{v,u} \gamma_u^{2,v}(x) \|$$

$$\le \alpha \sum_{v \in C^1} |\gamma_v^1(x)| \| x - h_{\gamma^{2,v}, C^{2,v}}(x) \| + \beta \sum_{v \in C^1} |\gamma_v^1(x)| \| x - h_{\gamma^{2,v}, C^{2,v}}(x) \|^2$$

$$+ \beta \sum_{v \in C^1} |\gamma_v^1(x)| \sum_{u \in C^{2,v}} |\gamma_u^{2,v}(x)| \| u - h_{\gamma^{2,v}, C^{2,v}}(x) \|^2, \tag{3}$$

*where $w_{v,u} = f(u)$ for $u \in C^{2,v}$.*

Similar to the interpretation of Lemma 2.1, bounds in Lemma 2.2 implies that we can approximate a nonlinear function $f(x)$ with linear function of the form

$$\sum_{v \in C^1} w_v \gamma_v^1(x) + \sum_{v \in C^1} \sum_{u \in C^{2,v}} w_{v,u} \gamma_v^1(x) \gamma_u^{2,v}(x),$$

where $[w_v]$ and $[w_{v,u}]$ are the unknown linear coefficients to be learned, and $[\gamma_v^1(x)]_{v \in C^1}$ and $[\gamma_u^{2,v}(x)]_{v \in C^1, u \in C^{2,v}}$ form the feature vector. The coding can be learned from unlabeled data by minimizing the right hand side of (2) or (3).

Compare with the single-layer coding, we note that the second term on the right hand side of (1) is replaced by the third term on the right hand side of (2). That is, the linear approximation power of the single-layer coding scheme (with a quadratic error term) becomes quadratic approximation power of the two-layer coding scheme (with a cubic error term). The first term on the right hand side of (1) is replaced by the first two terms on the right hand of (2). If the manifold is relatively flat, then the error terms $\|x - h_{\gamma^1, C^1}(x)\|$ and $\|x - h_{\gamma^{2,v}, C^{2,v}}(x)\|$ will be relatively small in comparison to the second term on the right hand side of (1). In such case the two-layer coding scheme can potentially improve the single-layer system significantly. This result is similar to that of [16], where the second layer uses local PCA instead of another layer of nonlinear coding. However, the bound in Lemma 2.2 is more refined and specifically applicable to nonlinear coding. The bound in (2) shows the potential of the two-layer coding scheme in achieving higher order approximation power than single layer coding. Higher order approximation gives meaningful improvement when each $|C^{2,v}|$ is relatively small compared to $|C^1|$. On the other hand, if $|C^1|$ is small but each $|C^{2,v}|$ is relatively large, then achieving higher order approximation does not lead to meaningful improvement. In such case, the bound in (3) shows that the performance of the two-level coding is still comparable to that of one-level coding scheme in (1). This is the situation where the 1st layer is mainly used to partition the space (while its approximation accuracy is not important), while the main approximation power is achieved with the second layer. The main advantage of two-layer coding in this case is to save computation. This is because instead of solving a single layer coding system with many parameters, we can solve many smaller coding systems, each with a small number of parameters. This is the situation when including nonlinearity in the second layer becomes useful, which means that the deep-coding network approach in this paper has some advantage over [16] which can only approximate linear function with local PCA in the second layer.

## 3 Deep Coding Network

We shall discuss the computational algorithm motivated by Lemma 2.2. While the two bounds (2) and (3) consider different scenarios depending on the relative size of the first layer versus the second layer, in reality it is difficult to differentiate and usually both bounds play a role at the same time. Therefore we have to consider a mixed effect. Instead of minimizing one bound versus another, we shall use them to motivate our algorithm, and design a method that accommodate the underlying intuition reflected by the two bounds.

### 3.1 Two Layer Formulation

In the following, we let $C^1 = \{v_1, \ldots, v_{L_1}\}$, $\gamma_{v_j}^1(X_i) = \gamma_j^i$, $C^{2,v_j} = \{v_{j,1}, \ldots, v_{j,L_2}\}$, and $\gamma_{v_{j,k}}^{2,v_j}(X_i) = \gamma_{j,k}^i$, where $L_1$ is the size of the first-layer codebook, and $L_2$ is the size of each individual codebook at the second layer. We take a layer-by-layer approach for training, where the second layer is regarded as a refinement of the first layer, which is consistent with Lemma 2.2. In the first layer, we learn a simple sparse coding model with all data:

$$[\gamma^1, C^1] = \arg\min_{\gamma, v} \left[ \sum_{i=1}^n \left( \frac{1}{2} \left\| X_i - \sum_{j=1}^{L_1} \gamma_j^i v_j \right\|_2^2 \right) \right]$$

$$\text{subject to } \gamma_j^i \geq 0, \sum_j \gamma_j^i = 1, \|v_j\| \leq \kappa, \tag{4}$$

where $\kappa$ is some constant, e.g., if all $X_i$ are normalized to have unit length, $\kappa$ can be set to be 1. For convenience, we not only enforce sum-to-one-constraint on the sparse coefficients, but also

impose nonnegative constraints so that $\sum_j |\gamma_j^i| = \sum_j \gamma_j^i = 1$ for all $i$. This presents a probability interpretation of the data, and allow us to approximate the following term on the right hand sides of (2) and (3):

$$\sum_j \gamma_j^i \left\| X_i - \sum_{k=1}^{L_2} \gamma_{j,k}^i v_{j,k} \right\| \leq \left( \sum_j \gamma_j^i \left\| X_i - \sum_{k=1}^{L_2} \gamma_{j,k}^i v_{j,k} \right\|^2 \right)^{1/2}.$$

Note that neither sum to one or 1-norm regularization of coefficients is needed in the derivation of (2), while such constraints are needed in (3). This means additional constraints may hurt performance in the case of (2) although it may help in the case of (3). Since we don't know which case is the dominant effect, as a compromise we remove the sum-to-one constraint but put in 1-norm regularization which is tunable. We still keep the positivity constraint for interpretability. This leads to the following formulation for the second layer:

$$[\gamma^{2,v_j}, C^{2,v_j}] = \arg\min_{\gamma,v} \left[ \sum_{i=1}^n \gamma_j^i \left( \frac{1}{2} \left\| X_i - \sum_{k=1}^{L_2} \gamma_{j,k}^i v_{j,k} \right\|_2^2 + \lambda_2 \sum_{k=1}^{L_2} \gamma_{j,k}^i \right) \right]$$
$$\text{subject to } \gamma_{j,k}^i \geq 0, \|v_{j,k}\| \leq 1, \tag{5}$$

where $\lambda_2$ is a $l_1$-norm sparsity regularization parameter controlling the sparseness of solutions. With the codings on both layers, the sparse representation of $X_i$ is $\left[ s\gamma_j^i, \ \gamma_j^i[\gamma_{j,1}^i, \ \gamma_{j,2}^i, \ ..., \ \gamma_{j,L_2}^i] \right]_{j=1,...L_1}$ where $s$ is a scaling factor balances the coding from the two different layers.

### 3.2 Multi-layer Extension

The two-level coding scheme can be easily extended to the third and higher layers. For example, at the third layer, for each base $v_{j,k}$, the third-layer coding is to solve the following weighted optimization:

$$[\gamma_3^{j,k}, C_3^{j,k}] = \arg\min_{\gamma,v} \left[ \sum_{i=1}^n \gamma_{j,k}^i \left( \frac{1}{2} \left\| X_i - \sum_{l=1}^{L_3} \gamma_{j,k,l}^i v_{j,k,l} \right\|_2^2 + \lambda_3 \sum_l \gamma_{j,k,l}^i \right) \right]$$
$$\text{subject to } \gamma_{j,k,l}^i \geq 0, \|v_{j,k,l}\| \leq 1. \tag{6}$$

### 3.3 Optimization

The optimization problems in Equations (4) to (6) can be generally solved by alternating the following two steps: 1) given current cookbook estimation $v$, compute the optimal sparse coefficients $\gamma$; 2) given the new estimates of the sparse coefficients, optimize the cookbooks.

Step 1 requires solving an independent optimization problem for each data sample, and it can be computationally very expensive when there are many training examples. In such case, computational efficiency becomes an important issue. We developed some efficient algorithms for solving the optimizations problem in Step 1 by exploiting the fact that the solutions of the optimization problems are sparse. The optimization problem in Step 1 of (4) can be posed as a nonnegative quadratic programming problem with a single sum-to-one equality constraint. We employ an active set method for this problem that easily handles the constraints [4]. Most importantly, since the optimal solutions are very sparse, the active set method often gives the exact solution after a few dozen of iterations. The optimization problem in (5) contains only nonnegative constraints (but not the sum-to-one constraint), for which we employ a pathwise projected Newton (PPN) method [3] that optimizes a block of coordinates per iteration instead of one coordinate at a time in the active set method. As a result, in typical sparse coding settings (for example, in the experiments that we will present shortly in Section 4), the PPN method is able to give the exact solution of a median size (e.g. 2048 dimension) nonnegative quadratic programming problem in milliseconds.

Step 2 can be solved in its dual form, which is convex optimization with nonnegative constraints [9]. Since the dual problem contains only nonnegative constraints, we can still employ projected Newton method. It is known that the projected Newton method has superlinear convergence rate under

fairly mild conditions [3]. The computational cost in Step 2 is often negligible compared to the computational cost in Step 1 when the cookbook size is no more than a few thousand.

A significant advantage of the second layer optimization in our proposal is parallelization. As shown in (5), the second-layer sparse coding is decomposed into $L_1$ independent coding problems, and thus can be naturally parallelized. In our implementation, this is done through Hadoop.

## 4  Experiments

### 4.1  MNIST dataset

We first demonstrate the effectiveness of the proposed deep coding scheme on the popular MNIST benchmark data [1]. MNIST dataset consists of 60,000 training digits and 10,000 testing digits. In our experiments of deep coding network, the entire training set is used to learn the first-layer coding, with codebook of size 64. For each of the 64 bases in the first layer, a second-layer codebook was learned – the deep coding scheme presented in the paper ensures that the codebook learning can be done independently. We implemented a Hadoop parallel program that solved the 64 codebook learning tasks in about an hour – which would have taken 64 hours on single machine. This shows that easy parallelization is a very attractive aspect of the proposed deep coding scheme, especially for large scale problems.

Table 1 shows the performance of deep coding network on MNIST compared to some previous coding schemes. There are a number of interesting observations in these results. First, adding an extra layer yields significant improvement on classification; e.g. for $L_1 = 512$, the classification error rate for single layer LCC is $2.60\%$ [17] while extended LCC achieves $1.98\%$ [16] (the extended LCC method in [16] may also be regarded as a two layer method but the second layer is linear); the two-layer coding scheme here significantly improves the performance with classification error rate of $1.51\%$. Second, the two-layer coding is less prone to overfitting than its single-layer counterpart. In fact, for the single-layer coding, our experiment shows that further increasing the codebook size will cause overfitting (e.g., with $L_1 = 8192$, the classification error deteriorates to $1.78\%$). In contrast, the performance of two-layer coding still improves when the second-layer codebook is as large as 512 (and the total codebook size is $64 \times 512 = 32768$, which is very high-dimensional considering the total number of training data is only 60,000). This property is desirable especially when high-dimensional representation is preferred in the case of using sparse coding plus linear classifier for classifications.

Figure 1 shows some first-layer bases and their associated second-layer bases. We can see that the second-layer bases provide deeper details that helps to further explain their first layer parent basis; on the other hand, the parent first-layer basis provides an informative context for its child second-layer bases. For example, in the seventh row in Fig. 1 where the first-layer basis is like Digit 7, this basis can come from Digit 7, Digit 9 or even Digit 4. Then, its second-layer bases help to further explain the meaning of the first-layer basis: in its associated second-layer bases, the first two bases in that row are parts of Digit 9 while the last basis in that row is a part of Digit '4'. Meanwhile, the first-layer 7-like basis provides important context for its second-layer part-like bases – without the first-layer basis, the fragmented parts (like the first two second-layer bases in that row) may not be very informative. The zoomed-in details contained in deeper bases significantly help a classifier to resolve difficult examples, and interestingly, coarser details provide useful context for finer details.

| Single layer sparse coding | | | | |
|---|---|---|---|---|
| Number of bases ($L_1$) | 512 | 1024 | 2048 | 4096 |
| Local coordinate coding | 2.60 | 2.17 | 1.79 | 1.75 |
| Extended LCC | 1.95 | 1.82 | 1.78 | 1.64 |
| Two-layer sparse coding | | | | |
| Number of bases ($L_2$) | 64 | 128 | 256 | 512 |
| $L_1 = 64$ | 1.85 | 1.69 | 1.53 | 1.51 |

Table 1: The classification error rate (in %) on MNIST dataset with different sparse coding schemes.

First–layer bases          Second–layer bases

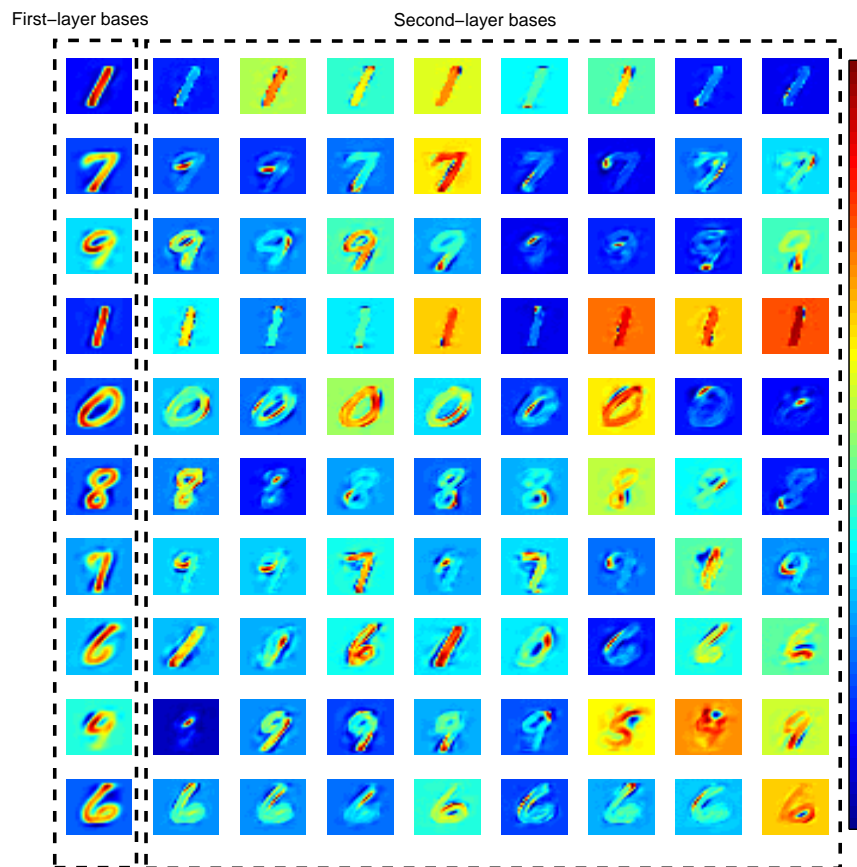

Figure 1: Example of bases from a two-layer coding network on MNIST data. For each row, the first image is a first-layer basis, and the remaining images are its associated second-layer bases. The colorbar is the same for all images, but the range it represents differs from image to image – generally, the color of the background of a image represent zero value, and the colors above and below that color respectively represent positive and negative values.

## 4.2   PASCAL 2007

The PASCAL 2007 dataset [6] consists of 20 categories of images such as airplanes, persons, cats, tables, and so on. It consists of 2501 training images and 2510 validation images, and the task is to classify an image into one or more of the 20 categories. Therefore, this task can be casted as training 20 binary classifiers. The critical issue is how to extract effective visual features from the images. Among different methods, one particularly effective approach is to use sparse coding to derive a codebook of low-level features (such as SIFT) and represent an image as a bag of visual words [15]. Here, we intend to learn two-layer hierarchical codebooks instead of single flat codebook for the bag-of-word image representation.

In our experiments, we first sampled dense SIFT descriptors (each is represented by a $128 \times 1$ vector) on each image using four scales, $7 \times 7$, $16 \times 16$, $25 \times 25$ and $31 \times 31$ with stepsize of 4. Then, the SIFT descriptors from all images (both training and validation images) were utilized to learn first-layer codebooks with different dimensions, $L_1 = 512, 1024$ and $2048$. Then, given a first-layer codebook, for each basis in the codebook, we learned its second-layer codebook of size 64 by solving the weighted optimization in (5). Again, the second-layer codebook learning was done in parallel using Hadoop. With the first-layer and second-layer codebooks, each SIFT feature was coded into a very high dimensional space: using $L_1 = 1024$ as an example, the coding dimension

| Dimension of the first layer ($L_1$) | 512 | 1024 | 2048 |
|---|---|---|---|
| Single-layer sparse coding | 42.7 | 45.3 | 48.4 |
| Two-layer sparse coding ($L_2$=64) | 51.1 | 52.8 | 53.3 |

Table 2: Average precision (in %) of classification on PASCAL07 dataset using different sparse coding schemes.

in total is $1024 + 1024 \times 64 = 66,560$. For each image, we employed $1 \times 1$, $2 \times 2$ and $1 \times 3$ spatial pyramid matching with max-pooling. Therefore in the end, each image is represented by a $532,480 (= 66,560 \times 8) \times 1$ high-dimensional vector for $L_1 = 1024$. Table 2 shows the classification results. It is clear that the two-layer sparse coding performs significantly better than its single-layer counterpart.

We would like to point out that, although we simply employed max-pooling in the experiments, it may not be the best pooling strategy for the hierarchical coding scheme presented in this paper. We believe a better pooling scheme needs to take the hierarchical structure into account, but this remains as an open problem and is one of our future work.

## 5 Conclusion

This paper proposes a principled extension of the traditional single-layer flat sparse coding scheme, where a two-layer coding scheme is derived based on theoretical analysis of nonlinear functional approximation that extends recent results for local coordinate coding. The two-layer approach can be easily generalized to deeper structures in a hierarchical multiple-layer manner. There are two main advantages of multi-layer coding: it can potentially achieve better performance because the deeper layers provide more details and structures; it is computationally more efficient because coding are decomposed into smaller problems. Experiment showed that the performance of two-layer coding can significantly improve that of single-layer coding.

For the future directions, it will be interesting to explore the deep coding network with more than two layers. The formulation proposed in this paper grants a straightforward extension from two layers to multiple layers. For small datasets like MNIST, the two-layer scheme seems to be already very powerful. However, for more complicated data, deeper coding with multiple layers may be an effective way for gaining finer and finer features. For example, the first layer coding picks up some large categories such as human, bikes, cups, and so on; then for the human category, the second-layer coding may find difference among adult, teenager, and senior person; and then the third layer may find even finer features such as race feature at different ages.

## References

[1] http://yann.lecun.com/exdb/mnist/.

[2] Samy Bengio, Fernando Pereira, Yoram Singer, and Dennis Strelow. Group sparse coding. In *NIPS' 09*, 2009.

[3] D P. Bertsekas. Projected newton methods for optimization problems with simple constraints. *SIAM J. Control Optim.*, 20(2):221–246, 1982.

[4] Dimitri P. Bertsekas. *Nonlinear programming*. Athena Scientific, 2003.

[5] David Bradley and J. Andrew (Drew) Bagnell. Differentiable sparse coding. In *Proceedings of Neural Information Processing Systems 22*, December 2008.

[6] M. Everingham, L. Van Gool, C. K. I. Williams, J. Winn, and A. Zisserman. The PASCAL Visual Object Classes Challenge 2007 (VOC2007) Results. http://www.pascal-network.org/challenges/VOC/voc2007/workshop/index.html.

[7] Mark Everingham. Overview and results of the classification challenge. *The PASCAL Visual Object Classes Challenge Workshop at ICCV*, 2009.

[8] G. E. Hinton and R. R. Salakhutdinov. Reducing the dimensionality of data with neural networks. *Science*, 313(5786):504 – 507, July 2006.

[9] Honglak Lee, Alexis Battle, Rajat Raina, and Andrew Y. Ng. Efficient sparse coding algorithms. In *Proceedings of the Neural Information Processing Systems (NIPS) 19*, 2007.

[10] Michael S. Lewicki and Terrence J. Sejnowski. Learning overcomplete representations. *Neural Computation*, 12:337–365, 2000.

[11] J. Mairal, F. Bach, J. Ponce, G. Sapiro, and A. Zisserman. Supervised dictionary learning. In *NIPS' 08*, 2008.

[12] B.A. Olshausen and D.J. Field. Emergence of simple-cell receptive field properties by learning a sparse code for nature images. *Nature*, 381:607–609, 1996.

[13] Rajat Raina, Alexis Battle, Honglak Lee, Benjamin Packer, and Andrew Y. Ng. Self-taught learning: Transfer learning from unlabeled data. *International Conference on Machine Learning*, 2007.

[14] Marc Aurelio Ranzato, Y-Lan Boureau, and Yann LeCun. Sparse feature learning for deep belief networks. In *NIPS' 07*, 2007.

[15] Jianchao Yang, Kai Yu, Yihong Gong, and Thomas Huang. Linear spatial pyramid matching using sparse coding for image classification. In *IEEE Conference on Computer Vision and Pattern Recognition*, 2009.

[16] Kai Yu and Tong Zhang. Improved local coordinate coding using local tangents. In *ICML' 09*, 2010.

[17] Kai Yu, Tong Zhang, and Yihong Gong. Nonlinear learning using local coordinate coding. In *NIPS' 09*, 2009.

